# Learning with Transformation Invariant Kernels

**Christian Walder**
Max Planck Institute for Biological Cybernetics
72076 Tübingen, Germany
`christian.walder@tuebingen.mpg.de`

**Olivier Chapelle**
Yahoo! Research
Santa Clara, CA
`chap@yahoo-inc.com`

## Abstract

This paper considers kernels invariant to translation, rotation and dilation. We show that no non-trivial positive definite (p.d.) kernels exist which are radial and dilation invariant, only conditionally positive definite (c.p.d.) ones. Accordingly, we discuss the c.p.d. case and provide some novel analysis, including an elementary derivation of a c.p.d. representer theorem. On the practical side, we give a support vector machine (s.v.m.) algorithm for arbitrary c.p.d. kernels. For the thin-plate kernel this leads to a classifier with only one parameter (the amount of regularisation), which we demonstrate to be as effective as an s.v.m. with the Gaussian kernel, even though the Gaussian involves a second parameter (the length scale).

## 1 Introduction

Recent years have seen widespread application of reproducing kernel Hilbert space (r.k.h.s.) based methods to machine learning problems (Schölkopf & Smola, 2002). As a result, kernel methods have been analysed to considerable depth. In spite of this, the aspects which we presently investigate seem to have received insufficient attention, at least within the machine learning community.

The first is transformation invariance of the kernel, a topic touched on in (Fleuret & Sahbi, 2003). Note we do not mean by this the local invariance (or insensitivity) of an algorithm to application specific transformations which should not affect the class label, such as one pixel image translations (see *e.g.* (Chapelle & Schölkopf, 2001)). Rather we are referring to global invariance to transformations, in the way that radial kernels (*i.e.* those of the form $k(\boldsymbol{x}, \boldsymbol{y}) = \phi(\|\boldsymbol{x} - \boldsymbol{y}\|)$) are invariant to translations. In Sections 2 and 3 we introduce the more general concept of transformation *scaledness*, focusing on translation, dilation and orthonormal transformations. An interesting result is that there exist no non-trivial p.d. kernel functions which are radial and dilation scaled.

There do exist non-trivial c.p.d. kernels with the stated invariances however. Motivated by this, we analyse the c.p.d. case in Section 4, giving novel elementary derivations of some key results, most notably a c.p.d. representer theorem. We then give in Section 6.1 an algorithm for applying the s.v.m. with arbitrary c.p.d. kernel functions. It turns out that this is rather useful in practice, for the following reason. Due to its invariances, the c.p.d. thin-plate kernel which we discuss in Section 5, is not only richly non-linear, but enjoys a duality between the length-scale parameter and the regularisation parameter of Tikhonov regularised solutions such as the s.v.m. In Section 7 we compare the resulting classifier (which has only a regularisation parameter), to that of the s.v.m. with Gaussian kernel (which has an additional length scale parameter). The results show that the two algorithms perform roughly as well as one another on a wide range of standard machine learning problems, notwithstanding the new method's advantage in having only one free parameter. In Section 8 we make some concluding remarks.

## 2 Transformation Scaled Spaces and Tikhonov Regularisation

**Definition 2.1.** Let $\mathcal{T}$ be a bijection on $\mathcal{X}$ and $\mathcal{F}$ a Hilbert space of functions on some non-empty set $\mathcal{X}$ such that $f \mapsto f \circ \mathcal{T}$ is a bijection on $\mathcal{F}$. $\mathcal{F}$ is $\mathcal{T}$-**scaled** if

$$\langle f, g \rangle_{\mathcal{F}} = g_{\mathcal{T}}\left(\mathcal{F}\right) \langle f \circ \mathcal{T}, g \circ \mathcal{T} \rangle_{\mathcal{F}} \tag{1}$$

for all $f \in \mathcal{F}$, where $g_{\mathcal{T}}\left(\mathcal{F}\right) \in \mathbb{R}^{+}$ is the *norm scaling function* associated with the operation of $\mathcal{T}$ on $\mathcal{F}$. If $g_{\mathcal{T}}\left(\mathcal{F}\right) = 1$ we say that $\mathcal{F}$ is $\mathcal{T}$-**invariant**.

The following clarifies the behaviour of Tikhonov regularised solutions in such spaces.

**Lemma 2.2.** *For any* $\Theta : \mathcal{F} \longrightarrow \mathbb{R}$ *and* $\mathcal{T}$ *such that* $f \mapsto f \circ \mathcal{T}$ *is a bijection of* $\mathcal{F}$, *if the left hand side is unique then*

$$\arg \min_{f \in \mathcal{F}} \Theta(f) = \left( \arg \min_{f_{\mathcal{T}} \in \mathcal{F}} \Theta(f_{\mathcal{T}} \circ \mathcal{T}) \right) \circ \mathcal{T}$$

*Proof.* Let $f^{*} = \arg \min_{f \in \mathcal{F}} \Theta(f)$ and $f_{\mathcal{T}}^{*} = \arg \min_{f_{\mathcal{T}} \in \mathcal{F}} \Theta(f_{\mathcal{T}} \circ \mathcal{T})$. By definition we have that $\forall g \in \mathcal{F}, \Theta(f_{\mathcal{T}}^{*} \circ \mathcal{T}) \leq \Theta(g \circ \mathcal{T})$. But since $f \mapsto f \circ \mathcal{T}$ is a bijection on $\mathcal{F}$, we also have $\forall g \in \mathcal{F}, \Theta(f_{\mathcal{T}}^{*} \circ \mathcal{T}) \leq \Theta(g)$. Hence, given the uniqueness, this implies $f^{*} = f_{\mathcal{T}}^{*} \circ \mathcal{T}$. □

The following Corollary follows immediately from Lemma 2.2 and Definition 2.1.

**Corollary 2.3.** *Let* $L_i$ *be any loss function. If* $\mathcal{F}$ *is* $\mathcal{T}$-*scaled and the left hand side is unique then*

$$\arg \min_{f \in \mathcal{F}} \left( \|f\|_{\mathcal{F}}^2 + \sum_i L_i \left(f\left(\boldsymbol{x}_i\right)\right) \right) = \left( \arg \min_{f \in \mathcal{F}} \left( \|f\|_{\mathcal{F}}^2 / g_{\mathcal{T}}\left(\mathcal{F}\right) + \sum_i L_i \left(f\left(\mathcal{T}\boldsymbol{x}_i\right)\right) \right) \right) \circ \mathcal{T}.$$

Corollary 2.3 includes various learning algorithms for various choices of $L_i$ — for example the s.v.m. with linear hinge loss for $L_i(t) = \max\left(0, 1 - y_i t\right)$, and kernel ridge regression for $L_i(t) = \left(y_i - t\right)^2$. Let us now introduce the specific transformations we will be considering.

**Definition 2.4.** Let $W_s$, $T_{\boldsymbol{a}}$ and $O_A$ be the **dilation, translation and orthonormal transformations** $\mathbb{R}^d \to \mathbb{R}^d$ defined for $s \in \mathbb{R} \setminus \{0\}$, $\boldsymbol{a} \in \mathbb{R}^d$ and orthonormal $A : \mathbb{R}^d \to \mathbb{R}^d$ by $W_s \boldsymbol{x} = s\boldsymbol{x}$, $T_{\boldsymbol{a}} \boldsymbol{x} = \boldsymbol{x} + \boldsymbol{a}$ and $O_A \boldsymbol{x} = A\boldsymbol{x}$ respectively.

Hence, for an r.k.h.s. which is $W_s$-scaled for arbitrary $s \neq 0$, training an s.v.m. and dilating the resultant decision function by some amount is equivalent training the s.v.m. on similarly dilated input patterns but with a regularisation parameter adjusted according to Corollary 2.3.

While (Fleuret & Sahbi, 2003) demonstrated this phenomenon for the s.v.m. with a particular kernel, as we have just seen it is easy to demonstrate for the more general Tikhonov regularisation setting with any function norm satisfying our definition of transformation scaledness.

## 3 Transformation Scaled Reproducing Kernel Hilbert Spaces

We now derive the necessary and sufficient conditions for a reproducing kernel (r.k.) to correspond to an r.k.h.s. which is $\mathcal{T}$-scaled. The relationship between $\mathcal{T}$-scaled r.k.h.s.'s and their r.k.'s is easy to derive given the uniqueness of the r.k. (Wendland, 2004). It is given by the following novel

**Lemma 3.1** (Transformation scaled r.k.h.s.). *The r.k.h.s.* $\mathcal{H}$ *with r.k.* $k : \mathcal{X} \times \mathcal{X} \to \mathbb{R}$, *i.e. with* $k$ *satisfying*

$$\langle k(\cdot, \boldsymbol{x}), f(\cdot) \rangle_{\mathcal{H}} = f(\boldsymbol{x}), \tag{2}$$

*is* $\mathcal{T}$-*scaled iff*

$$k(\boldsymbol{x}, \boldsymbol{y}) = g_{\mathcal{T}}\left(\mathcal{H}\right) k(\mathcal{T}\boldsymbol{x}, \mathcal{T}\boldsymbol{y}). \tag{3}$$

Which we prove in the accompanying technical report (Walder & Chapelle, 2007) . It is now easy to see that, for example, the homogeneous polynomial kernel $k(\boldsymbol{x}, \boldsymbol{y}) = \langle \boldsymbol{x}, \boldsymbol{y} \rangle^p$ corresponds to a $W_s$-scaled r.k.h.s. $\mathcal{H}$ with $g_{W_s}(\mathcal{H}) = \langle \boldsymbol{x}, \boldsymbol{y} \rangle^p / \langle s\boldsymbol{x}, s\boldsymbol{y} \rangle^p = s^{-2p}$. Hence when the homogeneous polynomial kernel is used with the *hard-margin* s.v.m. algorithm, the result is invariant to multiplicative scaling of the training and test data. If the soft-margin s.v.m. is used however, then the invariance

holds only under appropriate scaling (as per Corollary 2.3) of the margin softness parameter (*i.e.* $\lambda$ of the later equation (14)).

We can now show that there exist no non-trivial r.k.h.s.'s with radial kernels that are also $W_s$-scaled for all $s \neq 0$. First however we need the following standard result on homogeneous functions:

**Lemma 3.2.** *If $\phi : [0, \infty) \to \mathbb{R}$ and $g : (0, \infty) \to \mathbb{R}$ satisfy $\phi(r) = g(s)\phi(rs)$ for all $r \geq 0$ and $s > 0$ then $\phi(r) = a\delta(r) + br^p$ and $g(s) = s^{-p}$, where $a, b, p \in \mathbb{R}$, $p \neq 0$, and $\delta$ is Dirac's function.*

Which we prove in the accompanying technical report (Walder & Chapelle, 2007). Now, suppose that $\mathcal{H}$ is an r.k.h.s. with r.k. $k$ on $\mathbb{R}^d \times \mathbb{R}^d$. If $\mathcal{H}$ is $T_{\boldsymbol{a}}$-invariant for all $\boldsymbol{a} \in \mathbb{R}^d$ then

$$k(\boldsymbol{x}, \boldsymbol{y}) = k(T_{-\boldsymbol{y}}\boldsymbol{x}, T_{-\boldsymbol{y}}\boldsymbol{y}) = k(\boldsymbol{x} - \boldsymbol{y}, \boldsymbol{0}) \triangleq \phi_T(\boldsymbol{x} - \boldsymbol{y}).$$

If in addition to this $\mathcal{H}$ is $O_A$-invariant for all orthogonal $A$, then by choosing $A$ such that $A(\boldsymbol{x}-\boldsymbol{y}) = \|\boldsymbol{x} - \boldsymbol{y}\| \, \hat{e}$ where $\hat{e}$ is an arbitrary unit vector in $\mathbb{R}^d$ we have

$$k(\boldsymbol{x}, \boldsymbol{y}) = k(O_A\boldsymbol{x}, O_A\boldsymbol{y}) = \phi_T(O_A(\boldsymbol{x} - \boldsymbol{y})) = \phi_T(\|\boldsymbol{x} - \boldsymbol{y}\| \, \hat{e}) \triangleq \phi_{OT}(\|\boldsymbol{x} - \boldsymbol{y}\|)$$

*i.e.* $k$ is radial. All of this is straightforward, and a similar analysis can be found in (Wendland, 2004). Indeed the widely used Gaussian kernel satisfies both of the above invariances. But if we now also assume that $\mathcal{H}$ is $W_s$-scaled for all $s \neq 0$ — this time with arbitrary $g_{W_s}(\mathcal{H})$ — then

$$k(\boldsymbol{x}, \boldsymbol{y}) = g_{W_s}(\mathcal{H})k(W_s\boldsymbol{x}, W_s\boldsymbol{y}) = g_{W_{|s|}}(\mathcal{H})\phi_{OT}(|s| \, \|\boldsymbol{x} - \boldsymbol{y}\|)$$

so that letting $r = \|\boldsymbol{x} - \boldsymbol{y}\|$ we have that $\phi_{OT}(r) = g_{W_{|s|}}(\mathcal{H})\phi_{OT}(|s| \, r)$ and hence by Lemma 3.2 that $\phi_{OT}(r) = a\delta(r) + br^p$ where $a, b, p \in \mathbb{R}$. This is positive semi-definite for the trivial case $p = 0$, but there are various ways of showing this cannot be non-trivially positive semi-definite for $p \neq 0$. One simple way is to consider two arbitrary vectors $\boldsymbol{x}_1$ and $\boldsymbol{x}_2$ such that $\|\boldsymbol{x}_1 - \boldsymbol{x}_2\| = d > 0$. For the corresponding Gram matrix

$$K \triangleq \begin{pmatrix} a & bd^p \\ bd^p & a \end{pmatrix},$$

to be positive semi definite we require $0 \leq \det(K) = a^2 - b^2 d^{2p}$, but for arbitrary $d > 0$ and $a < \infty$, this implies $b = 0$. This may seem disappointing, but fortunately there do exist c.p.d. kernel functions with the stated properties, such as the thin-plate kernel. We discuss this case in detail in Section 5, after the following particularly elementary and in part novel introduction to c.p.d. kernels.

# 4 Conditionally Positive Definite Kernels

In the last Section we alluded to c.p.d. kernel functions – these are given by the following

**Definition 4.1.** A continuous function $\phi : \mathcal{X} \times \mathcal{X} \to \mathbb{R}$ is **conditionally positive definite** with respect to (w.r.t.) the linear space of functions $\mathcal{P}$ if, for all $m \in \mathbb{N}$, all $\{\boldsymbol{x}_i\}_{i=1...m} \subset \mathcal{X}$, and all $\boldsymbol{\alpha} \in \mathbb{R}^m \setminus \{\boldsymbol{0}\}$ satisfying $\sum_{j=1}^m \alpha_j p(\boldsymbol{x}_j) = 0$ for all $p \in \mathcal{P}$, the following holds

$$\sum_{j,k=1}^m \alpha_j \alpha_k \phi(\boldsymbol{x}_j, \boldsymbol{x}_k) > 0. \tag{4}$$

Due to the positivity condition (4) — as opposed one of non negativity — we are referring to c.p.d. rather than conditionally positive *semi*-definite kernels. The c.p.d. case is more technical than the p.d. case. We provide a minimalistic discussion here — for more details we recommend *e.g.* (Wendland, 2004). To avoid confusion, let us note in passing that while the above definition is quite standard (see *e.g.* (Wendland, 2004; Wahba, 1990)), many authors in the machine learning community use a definition of c.p.d. kernels which corresponds to our definition when $\mathcal{P} = \{1\}$ (*e.g.* (Schölkopf & Smola, 2002)) or when $\mathcal{P}$ is taken to be the space of polynomials of some fixed maximum degree (*e.g.* (Smola et al., 1998)). Let us now adopt the notation $\mathcal{P}^\perp(\boldsymbol{x}_1, \ldots, \boldsymbol{x}_m)$ for the set

$$\{\boldsymbol{\alpha} \in \mathbb{R}^m : \sum_{i=1}^m \alpha_i p(\boldsymbol{x}_i) = 0 \text{ for all } p \in \mathcal{P}\}.$$

The c.p.d. kernels of Definition 4.1 naturally define a Hilbert space of functions as per

**Definition 4.2.** Let $\phi : \mathcal{X} \times \mathcal{X} \to \mathbb{R}$ be a c.p.d. kernel w.r.t. $\mathcal{P}$. We define $F_\phi(\mathcal{X})$ to be the Hilbert space of functions which is the completion of the set

$$\left\{ \sum_{j=1}^m \alpha_j \phi(\cdot, \boldsymbol{x}_j) : m \in \mathbb{N}, \boldsymbol{x}_1, .., \boldsymbol{x}_m \in \mathcal{X}, \boldsymbol{\alpha} \in \mathcal{P}^\perp(\boldsymbol{x}_1, .., \boldsymbol{x}_m) \right\},$$

which due to the definition of $\phi$ we may endow with the inner product

$$\left\langle \sum_{j=1}^m \alpha_j \phi(\cdot, \boldsymbol{x}_j), \sum_{k=1}^n \beta_k \phi(\cdot, \boldsymbol{y}_k) \right\rangle_{F_\phi(\mathcal{X})} = \sum_{j=1}^m \sum_{k=1}^n \alpha_j \beta_k \phi(\boldsymbol{x}_j, \boldsymbol{y}_k). \tag{5}$$

Note that $\phi$ is not the r.k. of $F_\phi(\mathcal{X})$ — in general $\phi(\boldsymbol{x}, \cdot)$ does not even lie in $F_\phi(\mathcal{X})$. For the remainder of this Section we develop a c.p.d. analog of the representer theorem. We begin with

**Lemma 4.3.** *Let $\phi : \mathcal{X} \times \mathcal{X} \to \mathbb{R}$ be a c.p.d. kernel w.r.t. $\mathcal{P}$ and $p_1, \ldots p_r$ a basis for $\mathcal{P}$. For any $\{(\boldsymbol{x}_1, y_1), \ldots (\boldsymbol{x}_m, y_m)\} \subset \mathcal{X} \times \mathbb{R}$, there exists an $s = s_{F_\phi(\mathcal{X})} + s_\mathcal{P}$ where $s_{F_\phi(\mathcal{X})} = \sum_{j=1}^m \alpha_j \phi(\cdot, \boldsymbol{x}_j) \in F_\phi(\mathcal{X})$ and $s_\mathcal{P} = \sum_{k=1}^r \beta_k p_k \in \mathcal{P}$, such that $s(\boldsymbol{x}_i) = y_i, i = 1 \ldots m$.*

A simple and elementary proof (which shows (17) is solvable when $\lambda = 0$), is given in (Wendland, 2004) and reproduced in the accompanying technical report (Walder & Chapelle, 2007). Note that although such an interpolating function $s$ always exists, it need not be unique. The distinguishing property of the interpolating function is that the norm of the part which lies in $F_\phi(\mathcal{X})$ is minimum.

**Definition 4.4.** Let $\phi : \mathcal{X} \times \mathcal{X} \to \mathbb{R}$ be a c.p.d. kernel w.r.t. $\mathcal{P}$. We use the notation $P_\phi(\mathcal{P})$ to denote the projection $F_\phi(\mathcal{X}) \oplus \mathcal{P} \to F_\phi(\mathcal{X})$.

Note that $F_\phi(\mathcal{X}) \oplus P_\phi(\mathcal{P})$ is a direct sum since $p = \sum_{j=1}^m \beta_i \phi(\boldsymbol{z}_j, \cdot) \in \mathcal{P} \cap F_\phi(\mathcal{X})$ implies

$$\|p\|^2_{F_\phi(\mathcal{X})} = \langle p, p \rangle_{F_\phi(\mathcal{X})} = \sum_{i=1}^m \sum_{j=1}^n \beta_i \beta_j \phi(\boldsymbol{z}_i, \boldsymbol{z}_j) = \sum_{j=1}^m \beta_j p(\boldsymbol{z}_j) = 0.$$

Hence, returning to the main thread, we have the following lemma — our proof of which seems to be novel and particularly elementary.

**Lemma 4.5.** *Denote by $\phi : \mathcal{X} \times \mathcal{X} \to \mathbb{R}$ a c.p.d. kernel w.r.t. $\mathcal{P}$ and by $p_1, \ldots p_r$ a basis for $\mathcal{P}$. Consider an arbitrary function $s = s_{F_\phi(\mathcal{X})} + s_\mathcal{P}$ with $s_{F_\phi(\mathcal{X})} = \sum_{j=1}^m \alpha_j \phi(\cdot, \boldsymbol{x}_j) \in F_\phi(\mathcal{X})$ and $s_\mathcal{P} = \sum_{k=1}^r \beta_k p_k \in \mathcal{P}$. $\|P_\phi(\mathcal{P})s\|_{F_\phi(\mathcal{X})} \le \|P_\phi(\mathcal{P})f\|_{F_\phi(\mathcal{X})}$ holds for all $f \in F_\phi(\mathcal{X}) \oplus \mathcal{P}$ satisfying*

$$f(\boldsymbol{x}_i) = s(\boldsymbol{x}_i), \quad i = 1 \ldots m. \tag{6}$$

*Proof.* Let $f$ be an arbitrary element of $F_\phi(\mathcal{X}) \oplus \mathcal{P}$. We can always write $f$ as

$$f = \sum_{j=1}^m (\alpha_i + \overline{\alpha}_i) \phi(\cdot, \boldsymbol{x}_j) + \sum_{l=1}^n b_l \phi(\cdot, \boldsymbol{z}_l) + \sum_{k=1}^r c_k p_k.$$

If we define[1] $[P_x]_{i,j} = p_j(\boldsymbol{x}_i)$, $[P_z]_{i,j} = p_j(\boldsymbol{z}_i)$, $[\Phi_{xx}]_{i,j} = \phi(\boldsymbol{x}_i, \boldsymbol{x}_j)$, $[\Phi_{xz}]_{i,j} = \phi(\boldsymbol{x}_i, \boldsymbol{z}_j)$, and $[\Phi_{zx}]_{i,j} = \phi(\boldsymbol{z}_i, \boldsymbol{x}_j)$, then the condition (6) can hence be written

$$P_x \boldsymbol{\beta} = \Phi_{xx} \overline{\boldsymbol{\alpha}} + \Phi_{xz} \boldsymbol{b} + P_x \boldsymbol{c}, \tag{7}$$

and the definition of $F_\phi(\mathcal{X})$ requires that *e.g.* $\boldsymbol{\alpha} \in \mathcal{P}^\perp(\boldsymbol{x}_1, \ldots, \boldsymbol{x}_m)$, hence implying the constraints

$$P_x^\top \boldsymbol{\alpha} = \boldsymbol{0} \quad \text{and} \quad P_x^\top (\boldsymbol{\alpha} + \overline{\boldsymbol{\alpha}}) + P_z^\top \boldsymbol{b} = \boldsymbol{0}. \tag{8}$$

The inequality to be demonstrated is then

$$L \triangleq \boldsymbol{\alpha}^\top \Phi_{xx} \boldsymbol{\alpha} \le \begin{pmatrix} \boldsymbol{\alpha} + \overline{\boldsymbol{\alpha}} \\ \boldsymbol{b} \end{pmatrix}^\top \underbrace{\begin{pmatrix} \Phi_{xx} & \Phi_{xz} \\ \Phi_{zx} & \Phi_{zz} \end{pmatrix}}_{\triangleq \Phi} \begin{pmatrix} \boldsymbol{\alpha} + \overline{\boldsymbol{\alpha}} \\ \boldsymbol{b} \end{pmatrix} \triangleq R. \tag{9}$$

By expanding

$$R = \underbrace{\boldsymbol{\alpha}^\top \Phi_{xx} \boldsymbol{\alpha}}_{=L} + \underbrace{\begin{pmatrix} \overline{\boldsymbol{\alpha}} \\ \boldsymbol{b} \end{pmatrix}^\top \Phi \begin{pmatrix} \overline{\boldsymbol{\alpha}} \\ \boldsymbol{b} \end{pmatrix}}_{\triangleq \Delta_1} + 2 \underbrace{\begin{pmatrix} \boldsymbol{\alpha} \\ \boldsymbol{0} \end{pmatrix}^\top \Phi \begin{pmatrix} \overline{\boldsymbol{\alpha}} \\ \boldsymbol{b} \end{pmatrix}}_{\triangleq \Delta_2},$$

it follows from (8) that $P_x^\top \overline{\boldsymbol{\alpha}} + P_z^\top \boldsymbol{\beta} = \boldsymbol{0}$, and since $\Phi$ is c.p.d. w.r.t. $\begin{pmatrix} P_x^\top & P_z^\top \end{pmatrix}$ that $\Delta_1 \ge 0$. But (7) and (8) imply that $L \le R$, since

$$\Delta_2 = \boldsymbol{\alpha}^\top \Phi_{xx} \overline{\boldsymbol{\alpha}} + \boldsymbol{\alpha}^\top \Phi_{xz} \boldsymbol{b} = \overbrace{\boldsymbol{\alpha}^\top P_x}^{=0} (\boldsymbol{\beta} - \boldsymbol{c}) - \boldsymbol{\alpha}^\top \Phi_{xz} \boldsymbol{b} + \boldsymbol{\alpha}^\top \Phi_{xz} \boldsymbol{b} = 0. \qquad \square$$

Using these results it is now easy to prove an analog of the representer theorem for the p.d. case.

**Theorem 4.6** (Representer theorem for the c.p.d. case)**.** *Denote by $\phi : \mathcal{X} \times \mathcal{X} \to \mathbb{R}$ a c.p.d. kernel w.r.t. $\mathcal{P}$, by $\Omega$ a strictly monotonic increasing real-valued function on $[0, \infty)$, and by $c : \mathbb{R}^m \to \mathbb{R} \cup \{\infty\}$ an arbitrary cost function. There exists a minimiser over $F_\phi(\mathcal{X}) \oplus \mathcal{P}$ of*

$$W(f) \triangleq c\left(f(\boldsymbol{x}_1), \dots, f(\boldsymbol{x}_m)\right) + \Omega\left(\|P_\phi(\mathcal{P})f\|^2_{F_\phi(\mathcal{X})}\right) \tag{10}$$

*which admits the form $\sum_{i=1}^m \alpha_i \phi(\cdot, \boldsymbol{x}_i) + p$, where $p \in \mathcal{P}$.*

*Proof.* Let $f$ be a minimiser of W. Let $s = \sum_{i=1}^m \alpha_i \phi(\cdot, \boldsymbol{x}_i) + p$ satisfy $s(\boldsymbol{x}_i) = f(\boldsymbol{x}_i), i = 1 \dots m$. By Lemma 4.3 we know that such an $s$ exists. But by Lemma 4.5 $\|P_\phi(\mathcal{P})s\|^2_{F_\phi(\mathcal{X})} \geq \|P_\phi(\mathcal{P})f\|^2_{F_\phi(\mathcal{X})}$. As a result, $W(s) \leq W(f)$ and $s$ is a minimizer of W with the correct form. $\quad\square$

## 5 Thin-Plate Regulariser

**Definition 5.1.** The $m$-th order **thin-plate kernel** $\phi_m : \mathbb{R}^d \times \mathbb{R}^d \to \mathbb{R}$ is given by

$$\phi_m(\boldsymbol{x}, \boldsymbol{y}) = \begin{cases} (-1)^{m-(d-2)/2} \|\boldsymbol{x} - \boldsymbol{y}\|^{2m-d} \log(\|\boldsymbol{x} - \boldsymbol{y}\|) & \text{if } d \in 2\mathbb{N}, \\ (-1)^{m-(d-1)/2} \|\boldsymbol{x} - \boldsymbol{y}\|^{2m-d} & \text{if } d \in (2\mathbb{N} - 1), \end{cases} \tag{11}$$

for $\boldsymbol{x} \neq \boldsymbol{y}$, and zero otherwise. $\phi_m$ is c.p.d. with respect to $\pi_{m-1}(\mathbb{R}^d)$, the set of $d$-variate polynomials of degree at most $m - 1$. The kernel induces the following norm on the space $F_{\phi_m}(\mathbb{R}^d)$ of Definition 4.2 (this is not obvious — see *e.g.* (Wendland, 2004; Wahba, 1990))

$$\begin{aligned} \langle f, g \rangle_{F_{\phi_m}(\mathbb{R}^d)} & \triangleq \langle \psi f, \psi g \rangle_{L_2(\mathbb{R}^d)} \\ &= \sum_{i_1=1}^d \cdots \sum_{i_m=1}^d \int_{x_1=-\infty}^\infty \cdots \int_{x_d=-\infty}^\infty \left(\frac{\partial}{\partial x_{i_1}} \cdots \frac{\partial}{\partial x_{i_m}} f\right)\left(\frac{\partial}{\partial x_{i_1}} \cdots \frac{\partial}{\partial x_{i_m}} g\right) \mathrm{d}x_1 \dots \mathrm{d}x_d, \end{aligned}$$

where $\psi : F_{\phi_m}(\mathbb{R}^d) \to L_2(\mathbb{R}^d)$ is a *regularisation operator*, implicitly defined above.

Clearly $g_{O_A}(F_{\phi_m}(\mathbb{R}^d)) = g_{T_{\boldsymbol{a}}}(F_{\phi_m}(\mathbb{R}^d)) = 1$. Moreover, from the chain rule we have

$$\frac{\partial}{\partial x_{i_1}} \cdots \frac{\partial}{\partial x_{i_m}}(f \circ W_s) = s^m \left(\frac{\partial}{\partial x_{i_1}} \cdots \frac{\partial}{\partial x_{i_m}} f\right) \circ W_s \tag{12}$$

and therefore since $\langle f, g \rangle_{L_2(\mathbb{R}^d)} = s^d \langle f \circ W_s, g \circ W_s \rangle_{L_2(\mathbb{R}^d)}$ ,we can immediately write

$$\langle \psi(f \circ W_s), \psi(g \circ W_s) \rangle_{L_2(\mathbb{R}^d)} = s^{2m} \langle (\psi f) \circ W_s, (\psi g) \circ W_s \rangle_{L_2(\mathbb{R}^d)} = s^{2m-d} \langle \psi f, \psi g \rangle_{L_2(\mathbb{R}^d)} \tag{13}$$

so that $g_{W_s}(F_{\phi_m}(\mathbb{R}^d)) = s^{-(2m-d)}$. Note that although it may appear that this can be shown more easily using (11) and an argument similar to Lemma 3.1, the process is actually more involved due to the log factor in the first case of (11), and it is necessary to use the fact that the kernel is c.p.d. w.r.t. $\pi_{m-1}(\mathbb{R}^d)$. Since this is redundant and not central to the paper we omit the details.

## 6 Conditionally Positive Definite s.v.m.

In the Section 3 we showed that non-trivial kernels which are both radial and dilation scaled cannot be p.d. but rather only c.p.d. It is therefore somewhat surprising that the s.v.m. — one of the most widely used kernel algorithms — has been applied only with p.d. kernels, or kernels which are c.p.d. respect only to $\mathcal{P} = \{1\}$ (see *e.g.* (Boughorbel et al., 2005)). After all, it seems interesting to construct a classifier independent not only of the absolute positions of the input data, but also of their absolute multiplicative scale.

Hence we propose using the thin-plate kernel with the s.v.m. by minimising the s.v.m. objective over the space $F_\phi(\mathcal{X}) \oplus \mathcal{P}$ (or in some cases just over $F_\phi(\mathcal{X})$, as we shall see in Section 6.1). For this we require somewhat non-standard s.v.m. optimisation software. The method we propose seems simpler and more robust than previously mentioned solutions. For example, (Smola et al., 1998) mentions the numerical instabilities which may arise with the direct application of standard solvers.

| Dataset | Gaussian | Thin-Plate | dim/$n$ | | Dataset | Gaussian | Thin-Plate | dim/$n$ |
|---|---|---|---|---|---|---|---|---|
| banana | **10.567 (0.547)** | 10.667 (0.586) | 2/3000* | | image | 3.210 (0.504) | **1.867 (0.338)** | 18/2086 |
| breast | **26.574 (2.259)** | 28.026 (2.900) | 9/263 | | ringnm | **1.533 (0.229)** | 1.833 (0.200) | 20/3000* |
| diabetes | 23.578 (0.989) | **23.452 (1.215)** | 8/768 | | splice | 8.931 (0.640) | **8.651 (0.433)** | 60/2844 |
| flare | **36.143 (0.969)** | 38.190 (2.317) | 9/144 | | thyroid | 4.199 (1.087) | **3.247 (1.211)** | 5/215 |
| german | **24.700 (1.453)** | 24.800 (1.373) | 20/1000 | | twonm | **1.833 (0.194)** | 1.867 (0.254) | 20/3000* |
| heart | 17.407 (2.142) | **17.037 (2.290)** | 13/270 | | wavefm | 8.333 (0.378) | **8.233 (0.484)** | 21/3000 |

Table 1: Comparison of Gaussian and thin-plate kernel with the s.v.m. on the UCI data sets. Results are reported as "mean % classification error (standard error)". *dim* is the input dimension and $n$ the total number of data points. A star in the $n$ column means that more examples were available but we kept only a maximum of 2000 per class in order to reduce the computational burden of the extensive number of cross validation and model selection training runs (see Section 7). None of the data sets were linearly separable so we always used used the normal ($\beta$ unconstrained) version of the optimisation described in Section 6.1.

## 6.1 Optimising an s.v.m. with c.p.d. Kernel

It is simple to implement an s.v.m. with a kernel $\phi$ which is c.p.d. w.r.t. an arbitrary finite dimensional space of functions $\mathcal{P}$ by extending the primal optimisation approach of (Chapelle, 2007) to the c.p.d. case. The quadratic loss s.v.m. solution can be formulated as $\arg \min_{f \in F_\phi(\mathcal{X}) \oplus \mathcal{P}}$ of

$$\lambda \|P_\phi(\mathcal{P})f\|^2_{F_\phi(\mathcal{X})} + \sum_{i=1}^{n} \max(0, 1 - y_i f(\boldsymbol{x}_i))^2, \tag{14}$$

Note that for the second order thin-plate case we have $\mathcal{X} = \mathbb{R}^d$ and $\mathcal{P} = \pi_1(\mathbb{R}^d)$ (the space of constant and first order polynomials). Hence $\dim(\mathcal{P}) = d + 1$ and we can take the basis to be $p_j(\boldsymbol{x}) = [\boldsymbol{x}]_j$ for $j = 1 \dots d$ along with $p_{d+1} = 1$.

It follows immediately from Theorem 4.6 that, letting $p_1, p_2, \dots p_{\dim(\mathcal{P})}$ span $\mathcal{P}$, the solution to (14) is given by $f_{\mathrm{svm}}(\boldsymbol{x}) = \sum_{i=1}^{n} \alpha_i \phi(\boldsymbol{x}_i, \boldsymbol{x}) + \sum_{j=1}^{\dim(\mathcal{P})} \beta_j p_j(\boldsymbol{x})$. Now, if we consider only the margin violators — those vectors which (at a given step of the optimisation process) satisfy $y_i f(\boldsymbol{x}_i) < 1$, we can replace the $\max(0, \cdot)$ in (14) with $(\cdot)$. This is equivalent to making a local second order approximation. Hence by repeatedly solving in this way while updating the set of margin violators, we will have implemented a so-called Newton optimisation. Now, since

$$\|P_\phi(\mathcal{P})f_{\mathrm{svm}}\|^2_{F_\phi(\mathcal{X})} = \sum_{i,j=1}^{n} \alpha_i \alpha_j \phi(\boldsymbol{x}_i, \boldsymbol{x}_j), \tag{15}$$

the local approximation of the problem is, in $\boldsymbol{\alpha}$ and $\boldsymbol{\beta}$

$$\text{minimise } \lambda \boldsymbol{\alpha}^\top \Phi \boldsymbol{\alpha} + \|\Phi \boldsymbol{\alpha} + P \boldsymbol{\beta} - \boldsymbol{y}\|^2, \text{ subject to } P^\top \boldsymbol{\alpha} = \boldsymbol{0}, \tag{16}$$

where $[\Phi]_{i,j} = \phi(\boldsymbol{x}_i, \boldsymbol{x}_j)$, $[P]_{j,k} = p_k(\boldsymbol{x}_j)$, and we assumed for simplicity that all vectors violate the margin. The solution in this case is given by (Wahba, 1990)

$$\begin{pmatrix} \boldsymbol{\alpha} \\ \boldsymbol{\beta} \end{pmatrix} = \begin{pmatrix} \lambda I + \Phi & P^\top \\ P & \boldsymbol{0} \end{pmatrix}^{-1} \begin{pmatrix} \boldsymbol{y} \\ \boldsymbol{0} \end{pmatrix}. \tag{17}$$

In practice it is essential that one makes a change of variable for $\boldsymbol{\beta}$ in order to avoid the numerical problems which arise when $P$ is rank deficient or numerically close to it. In particular we make the $QR$ factorisation (Golub & Van Loan, 1996) $P^\top = QR$, where $Q^\top Q = I$ and $R$ is square. We then solve for $\boldsymbol{\alpha}$ and $\overline{\boldsymbol{\beta}} = R\boldsymbol{\beta}$. As a final step at the end of the optimisation process, we take the minimum norm solution of the system $\overline{\boldsymbol{\beta}} = R\boldsymbol{\beta}$, $\boldsymbol{\beta} = R^{\#}\overline{\boldsymbol{\beta}}$ where $R^{\#}$ is the pseudo inverse of $R$. Note that although (17) is standard for squared loss regression models with c.p.d. kernels, our use of it in optimising the s.v.m. is new. The precise algorithm is given in (Walder & Chapelle, 2007), where we also detail two efficient factorisation techniques, specific to the new s.v.m. setting. Moreover, the method we present in Section 6.2 deviates considerably further from the existing literature.

## 6.2 Constraining $\beta = 0$

Previously, if the data can be separated with only the $\mathcal{P}$ part of the function space — *i.e.* with $\boldsymbol{\alpha} = \mathbf{0}$ — then the algorithm will always do so regardless of $\lambda$. This is correct in that, since $\mathcal{P}$ lies in the null space of the regulariser $\|P_\phi(\mathcal{P})\cdot\|^2_{F_\phi(\mathcal{X})}$, such solutions minimise (14), but may be undesirable for various reasons. Firstly, the regularisation cannot be controlled via $\lambda$. Secondly, for the thin-plate, $\mathcal{P} = \pi_1(\mathbb{R}^d)$ and the solutions are simple linear separating hyperplanes. Finally, there may exist infinitely many solutions to (14). It is unclear how to deal with this problem — after all it implies that the regulariser is simply inappropriate for the problem at hand. Nonetheless we still wish to apply a (non-linear) algorithm with the previously discussed invariances of the thin-plate.

To achieve this, we minimise (14) as before, but over the space $F_\phi(\mathcal{X})$ rather than $F_\phi(\mathcal{X}) \oplus \mathcal{P}$. It is important to note that by doing so we can no longer invoke Theorem 4.6, the representer theorem for the c.p.d. case. This is because the solvability argument of Lemma 4.3 no longer holds. Hence we do not know the optimal basis for the function, which may involve infinitely many $\phi(\cdot, \boldsymbol{x})$ terms. The way we deal with this is simple — instead of minimising over $F_\phi(\mathcal{X})$ we consider only the finite dimensional subspace given by

$$\left\{ \sum_{j=1}^n \alpha_j \phi(\cdot, \boldsymbol{x}_j) : \boldsymbol{\alpha} \in \mathcal{P}^\perp(\boldsymbol{x}_1, \dots, \boldsymbol{x}_n) \right\},$$

where $\boldsymbol{x}_1, \dots \boldsymbol{x}_n$ are those of the original problem (14). The required update equation can be acquired in a similar manner as before. The closed form solution to the constrained quadratic programme is in this case given by (see (Walder & Chapelle, 2007))

$$\boldsymbol{\alpha} = -P_\perp \left( P_\perp^\top \left( \lambda \Phi + \Phi_{sx}^\top \Phi_{sx} \right) P_\perp \right)^{-1} P_\perp^\top \Phi_{sx}^\top \boldsymbol{y}_s \tag{18}$$

where $\Phi_{sx} = [\Phi]_{s,:}$, $s$ is the current set of margin violators and $P_\perp$ the null space of $P$ satisfying $PP_\perp = \mathbf{0}$. The precise algorithm we use to optimise in this manner is given in the accompanying technical report (Walder & Chapelle, 2007), where we also detail efficient factorisation techniques.

## 7 Experiments and Discussion

We now investigate the behaviour of the algorithms which we have just discussed, namely the thin-plate based s.v.m. with 1) the optimisation over $F_\phi(\mathcal{X}) \oplus \mathcal{P}$ as per Section 6.1, and 2) the optimisation over a subspace of $F_\phi(\mathcal{X})$ as per Section 6.2. In particular, we use the second method if the data is linearly separable, otherwise we use the first. For a baseline we take the Gaussian kernel $k(\boldsymbol{x}, \boldsymbol{y}) = \exp\left( -\|\boldsymbol{x} - \boldsymbol{y}\|^2 / (2\sigma^2) \right)$, and compare on real world classification problems.

**Binary classification (UCI data sets).** Table 1 provides numerical evidence supporting our claim that the thin-plate method is competitive with the Gaussian, in spite of it's having one less hyper parameter. The data sets are standard ones from the UCI machine learning repository. The experiments are extensive — the experiments on binary problems alone includes all of the data sets used in (Mika et al., 2003) plus two additional ones (*twonorm* and *splice*). To compute each error measure, we used five splits of the data and tested on each split after training on the remainder. For parameter selection, we performed five fold cross validation on the four-fifths of the data available for training each split, over an exhaustive search of the algorithm parameter(s) ($\sigma$ and $\lambda$ for the Gaussian and happily just $\lambda$ for the thin-plate). We then take the parameter(s) with lowest mean error and retrain on the entire four fifths. We ensured that the chosen parameters were well within the searched range by visually inspecting the cross validation error as a function of the parameters. Happily, for the thin-plate we needed to cross validate to choose only the regularisation parameter $\lambda$, whereas for the Gaussian we had to choose both $\lambda$ and the scale parameter $\sigma$. The discovery of an equally effective algorithm which has only one parameter is important, since the Gaussian is probably the most popular and effective kernel used with the s.v.m. (Hsu et al., 2003).

**Multi class classification (USPS data set).** We also experimented with the 256 dimensional, ten class USPS digit recognition problem. For each of the ten one *vs.* the rest models we used five fold cross validation on the 7291 training examples to find the parameters, retrained on the full training set, and labeled the 2007 test examples according to the binary classifier with maximum output. The Gaussian misclassified 88 digits (4.38%), and the thin-plate 85 (4.25%). Hence the Gaussian did not perform significantly better, in spite of the extra parameter.

**Computational complexity.** The normal computational complexity of the c.p.d. s.v.m. algorithm is the usual $O(n_{sv}^3)$ — cubic in the number of margin violators. For the $\boldsymbol{\beta} = 0$ variant (necessary only on linearly separable problems — presently only the USPS set) however, the cost is $O(n_b^2 n_{sv} + n_b^3)$, where $n_b$ is the number of basis functions in the expansion. For our USPS experiments we expanded on all $m$ training points, but if $n_{sv} \ll m$ this is inefficient and probably unnecessary. For example the final ten models (those with optimal parameters) of the USPS problem had around 5% margin violators, and so training each Gaussian s.v.m. took only $\sim 40s$ in comparison to $\sim 17$ minutes (with the use of various efficient factorisation techniques as detailed in the accompanying (Walder & Chapelle, 2007) ) for the thin-plate. By expanding on only 1500 randomly chosen points however, the training time was reduced to $\sim 4$ minutes while incurring only 88 errors — the same as the Gaussian. Given that for the thin-plate cross validation needs to be performed over one less parameter, even in this most unfavourable scenario of $n_{sv} \ll m$, the overall times of the algorithms are comparable. Moreover, during cross validation one typically encounters larger numbers of violators for some suboptimal parameter configurations, in which cases the Gaussian and thin-plate training times are comparable.

## 8    Conclusion

We have proven that there exist no non-trivial radial p.d. kernels which are dilation invariant (or more accurately, dilation *scaled*), but rather only c.p.d. ones. Such kernels have the advantage that, to take the s.v.m. as an example, varying the absolute multiplicative scale (or length scale) of the data has the same effect as changing the regularisation parameter — hence one needs model selection to chose only one of these, in contrast to the widely used Gaussian kernel for example.

Motivated by this advantage we provide a new, efficient and stable algorithm for the s.v.m. with arbitrary c.p.d. kernels. Importantly, our experiments show that the performance of the algorithm nonetheless matches that of the Gaussian on real world problems.

The c.p.d. case has received relatively little attention in machine learning. Our results indicate that it is time to redress the balance. Accordingly we provided a compact introduction to the topic, including some novel analysis which includes an new, elementary and self contained derivation of one particularly important result for the machine learning community, the representer theorem.

## Footnotes

[1] Square brackets w/ subscripts denote matrix elements, and colons denote entire rows or columns.

## References

Boughorbel, S., Tarel, J.-P., & Boujemaa, N. (2005). Conditionally positive definite kernels for svm based image recognition. *Proc. of IEEE ICME'05*. Amsterdam.

Chapelle, O. (2007). Training a support vector machine in the primal. *Neural Computation*, *19*, 1155–1178.

Chapelle, O., & Schölkopf, B. (2001). Incorporating invariances in nonlinear support vector machines. In T. Dietterich, S. Becker and Z. Ghahramani (Eds.), *Advances in neural information processing systems 14*, 609–616. Cambridge, MA: MIT Press.

Fleuret, F., & Sahbi, H. (2003). Scale-invariance of support vector machines based on the triangular kernel. *Proc. of ICCV SCTV Workshop*.

Golub, G. H., & Van Loan, C. F. (1996). *Matrix computations*. Baltimore MD: The Johns Hopkins University Press. 2nd edition.

Hsu, C.-W., Chang, C.-C., & Lin, C.-J. (2003). *A practical guide to support vector classification* (Technical Report). National Taiwan University.

Mika, S., Rätsch, G., Weston, J., Schölkopf, B., Smola, A., & Müller, K.-R. (2003). Constructing descriptive and discriminative non-linear features: Rayleigh coefficients in feature spaces. *IEEE PAMI*, *25*, 623–628.

Schölkopf, B., & Smola, A. J. (2002). *Learning with kernels: Support vector machines, regularization, optimization, and beyond*. Cambridge: MIT Press.

Smola, A., Schölkopf, B., & Müller, K.-R. (1998). The connection between regularization operators and support vector kernels. *Neural Networks*, *11*, 637–649.

Wahba, G. (1990). *Spline models for observational data*. Philadelphia: Series in Applied Math., Vol. 59, SIAM.

Walder, C., & Chapelle, O. (2007). *Learning with transformation invariant kernels* (Technical Report 165). Max Planck Institute for Biological Cybernetics, Department of Empirical Inference, Tübingen, Germany.

Wendland, H. (2004). *Scattered data approximation*. Monographs on Applied and Computational Mathematics. Cambridge University Press.

